# VLSI Phase Locking Architectures for Feature Linking in Multiple Target Tracking Systems

**Andreas G. Andreou**
andreou@jhunix.hcf.jhu.edu
Department of Electrical and
Computer Engineering
The Johns Hopkins University
Baltimore, MD 21218

**Thomas G. Edwards**
tedwards@src.umd.edu
Department of Electrical Engineering
The University of Maryland
College Park, MD 20722

## Abstract

Recent physiological research has shown that synchronization of oscillatory responses in striate cortex may code for relationships between visual features of objects. A VLSI circuit has been designed to provide rapid phase-locking synchronization of multiple oscillators to allow for further exploration of this neural mechanism. By exploiting the intrinsic random transistor mismatch of devices operated in subthreshold, large groups of phase-locked oscillators can be readily partitioned into smaller phase-locked groups. A multiple target tracker for binary images is described utilizing this phase-locking architecture. A VLSI chip has been fabricated and tested to verify the architecture. The chip employs Pulse Amplitude Modulation (PAM) to encode the output at the periphery of the system.

## 1 Introduction

In striate cortex, visual information coming from the retina (via the lateral geniculate nuclei) is processed to extract retinotopic maps of visual features. Some cells in cortex are receptive to lines of particular orientation, length, and/or movement direction (Hubel, 1988). A fundamental problem of visual processing is how to

associate certain groups of features together to form coherent representations of objects. Since there is an almost infinite number of possible feature combinations, it seems unlikely that there are dedicated "grandmother" cells which code for every possible feature combination. There probably exists a type of adaptive and transitory method to "bind" these features together. The *Binding Problem* (Crick, 1990) is the problem of making neural elements which are receptive to these visual features temporarily become active as a group that codes for a particular object, yet maintaining the group's specificity towards that object, even when there are several different interleaved objects in the visual field.

Temporal correlation of neural response is one solution to the binding problem (von der Malsburg, 1986). Response from neurons (or neural oscillating circuits) which are receptive to a particular visual feature are required to have high temporal correlation with responses to other visual features that correspond to the same object. This would require that there is stimulus-driven oscillation in visual cortex, and that there is also a degree of oscillation synchronization between neural circuits receptive to the same object. Both of these requirements have been found in visual cortex (Gray, 1987; Gray, 1989). Furthermore, there have been several computer simulations of the synchronization phenomena and related visual processing tasks (Baldi, 1990; Eckhorn, 1990).

This paper describes a phase-locking architecture for a circuit which performs a multiple-target tracking problem. It will accomplish this task by establishing a zero valued phase difference between oscillators that are receptive to those features to be "bound" together to form an object. Each object will then be recognized as a group of synchronous oscillators, and oscillators that correspond to different objects will be identified due to their lack of synchronization. We assume these oscillators have low duty-cycle pulsed outputs, and the oscillators which correspond to the same object will all pulse high at the same time. Target location will be communicated to the periphery by Pulse Amplitude Modulation (PAM).

## 2   The Neural Oscillator

The oscillator for the target tracker must have two qualities. It needs to be capable of producing a fairly smooth phase representation so that it is easy to compare the difference between oscillator phases to allow for robust phase-locking. It is also useful to have a pulsed output present so that one group of oscillators can be easily discerned from another group of oscillators when their outputs are examined over time. The self-resetting neuron circuit (Mead, 1989) provides both of these outputs (Figure 1). Current $I_{in}$ provided by FET Q1 charges capacitor C1 until positive feedback though the non-inverting CMOS amplifier and capacitor C2 brings $V_{phase}$ all the way to $V_{dd}$. This causes the output voltage to go high, which turns on Q2 thus draining charge from C1 by $I_{reset}$ through Q3 and lowering $V_{phase}$. When the $V_{phase}$ is brought low enough, positive feedback brings both $V_{phase}$ and the output voltage down to $V_{ss}$. This turns transistor Q2 off, and the cycle repeats. The duration of output pulses is inversely proportional to $I_{reset} - I_{in}$, and the time between output pulses is inversely proportional to $I_{in}$. Figure 2 is a plot of the pulse output voltage and $V_{phase}$ vs. time.

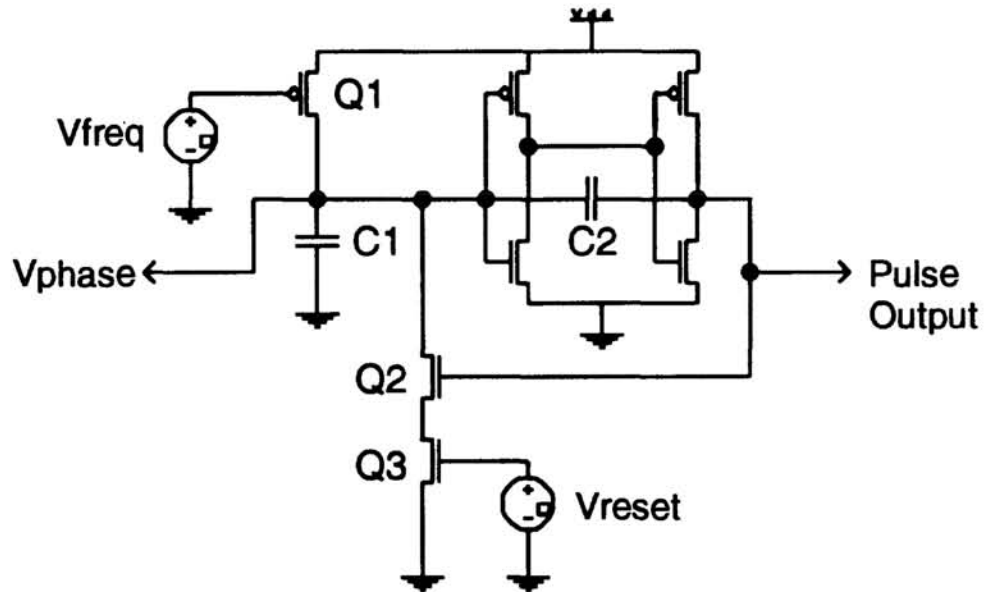

Figure 1: Self-Resetting Neural Oscillator

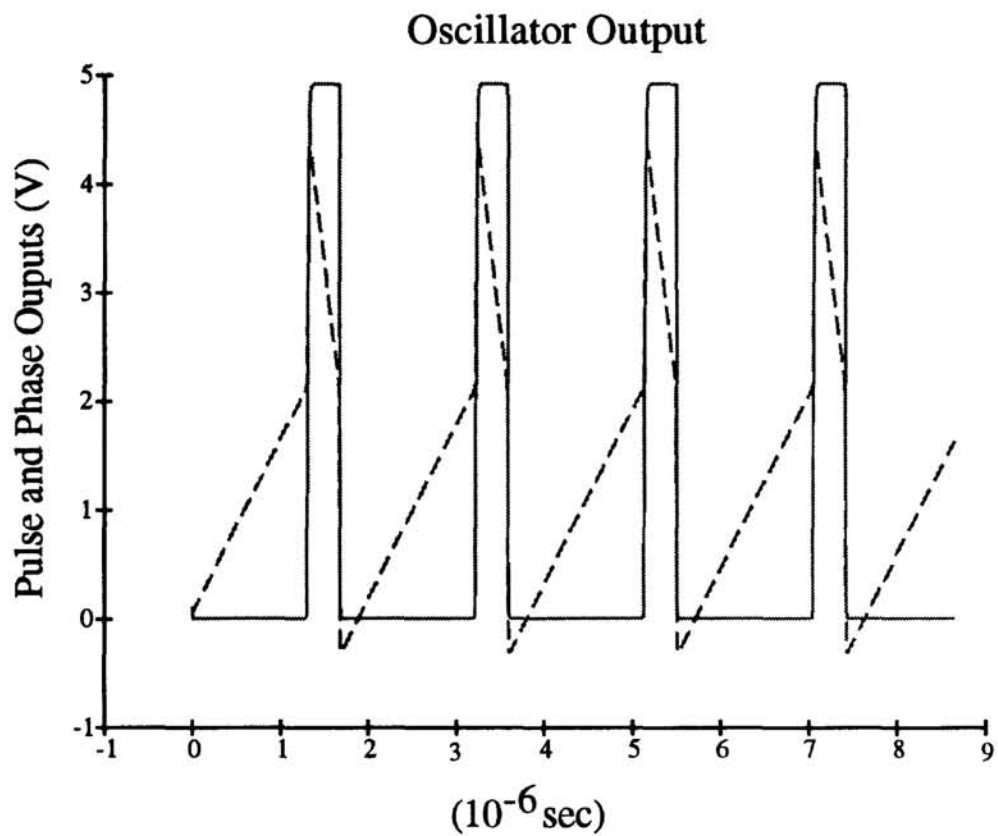

Figure 2: Plot of Pulse Output (line) and Phase Voltage (dashed) vs. Time for Neural Oscillator

## 3  Phase Locking

To achieve stable and reliable performance, the *Comparator Model* (Kammen, 1990) of phase-locking was used. Oscillator phase is adjusted according to

$$\frac{\partial \theta(x,t)}{\partial t} = \omega(x) + f\left(\frac{1}{n}\sum_{i=1}^{n}\theta(i,t) - \theta(x,t)\right).$$

Where $\theta(x,t)$ is the phase of oscillator $x$ at time $t$, $\omega(x)$ is the intrinsic phase advance of oscillator $x$, $n$ is the total number of oscillators, and $f$ is a sigmoid squashing-function.

Each object in the visual field requires one averaging circuit to achieve phase-locking of its receptive oscillators. But at any time we do not know the number of objects which will be in the visual field. Therefore, instead of having a pool of monolithic averaging circuits, it is preferable to distribute the averaging function over all the oscillator cells in a way which allows partitioning of the visual field into multiple phase-locked groups of oscillators. The follower-aggregator circuit (Mead, 1989) can be used to develop the average phase information using current-mode computation. It consists of transconductance amplifiers connected as voltage-followers with all outputs tied together to form the average of all input voltages.

The phase averaging circuitry can be distributed among the oscillators by placing one transconductance amplifier in each oscillator cell, and linking those oscillators to be phase-locked by a common line. The visual field can be partitioned into multiple phase-locked groups with separate average phases by using FETs to gate whether or not the averaging information can pass through an oscillator cell to its neighbors.

To lock an oscillator in phase with the rest of the oscillators which are attached to the averaging line, extra current is provided to the oscillator by a transconductance amplifier to slightly speed up or slow down the oscillator to match its phase to the average phase of the oscillators in the group. Figure 3 shows the circuit for a complete phase-locking oscillator cell.

Computer simulations of this phase-locking system were carried out using the Analog circuit simulator. Figure 4 shows the result of a simulation of two oscillators. $V_{gate}$ is the voltage controlling the NFET of the transmission gate which links the phase averaging lines of the two oscillators together (the PFETs are controlled complementary). As soon as the $V_{gate}$ is brought high, the oscillators rapidly phase lock.

## 4  Target Location

We will assume that the input to a visual tracking chip is a binary image projected onto the die. Phototransistors detect the brightness of each pixel, and if it is above a threshold level, the pixel control circuitry will turn the pixel's oscillator on. If a pixel oscillator is turned on, gating circuitry will allow the propagation of the phase averaging line through the pixel's oscillator cell to its nearest-neighbors. Illuminated

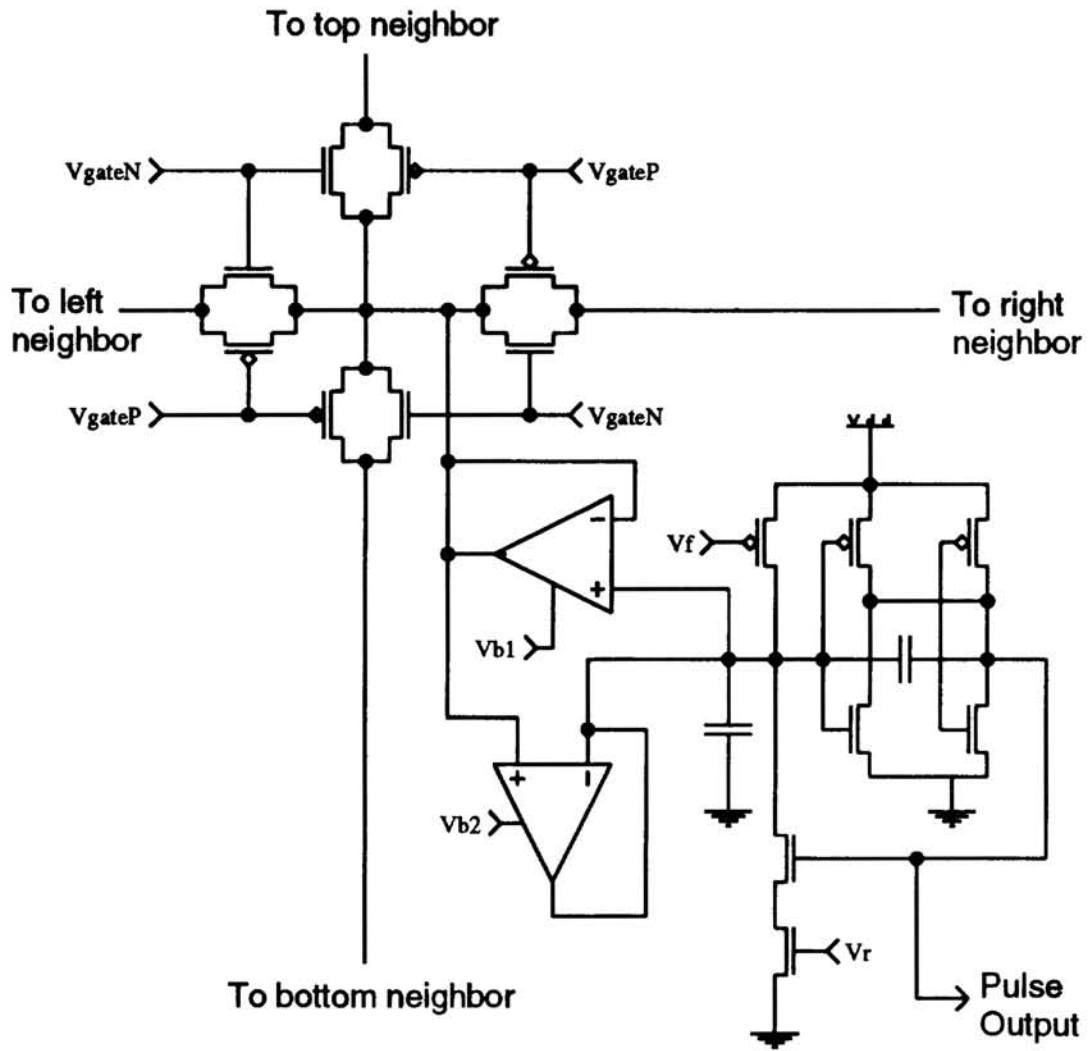

Figure 3: Phase-Locking Oscillator Cell

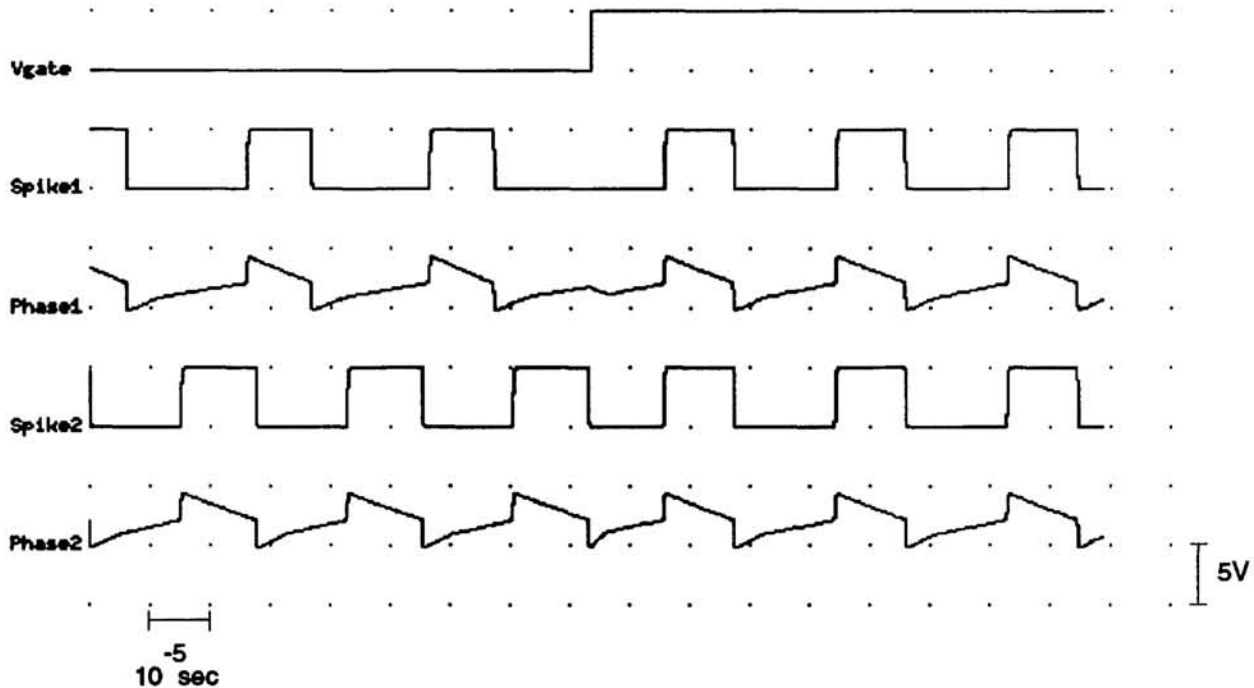

Figure 4: Phase-Locking Simulation

nearest-neighbor connected pixels will thus have their oscillators turned on and will become phase-locked.

The follower-aggregator circuit can be modified to determine linear position (Maher, 1989) by using voltage taps off of a resistive line as inputs to the transconductance amplifiers, and biasing the amplifiers by currents that correspond to the pulsing outputs of the oscillators (see Figure 5).

During the time that a group of oscillators are spiking, the output of the tracking circuitry will yield a location corresponding to the average position of the distribution of those oscillators. There can be many different nearest-neighbor connected objects projected onto the die, and the position of the center of each object is communicated to the periphery via PAM. Thus, we can use multiplexing in time to simplify connectivity of communication with the periphery of the chip.

## 5  Test Chip

A chip to test the Comparator Model phase-locking method and multiple-target tracking system was fabricated by the MOSIS service in 2.0 $\mu m$ feature size CMOS. To keep this test chip simple, the oscillators were arranged in a one-dimensional chain, and voltage inputs to the chip were used to control whether or not a pixel was considered "illuminated." A polysilicon resistive line was used to provide linear position information to the tracking system. All transistors used were minimum size (6 $\mu m$ wide and 4 $\mu m$ long).

The test chip was able to rapidly and robustly phase-lock groups of nearest-neighbor

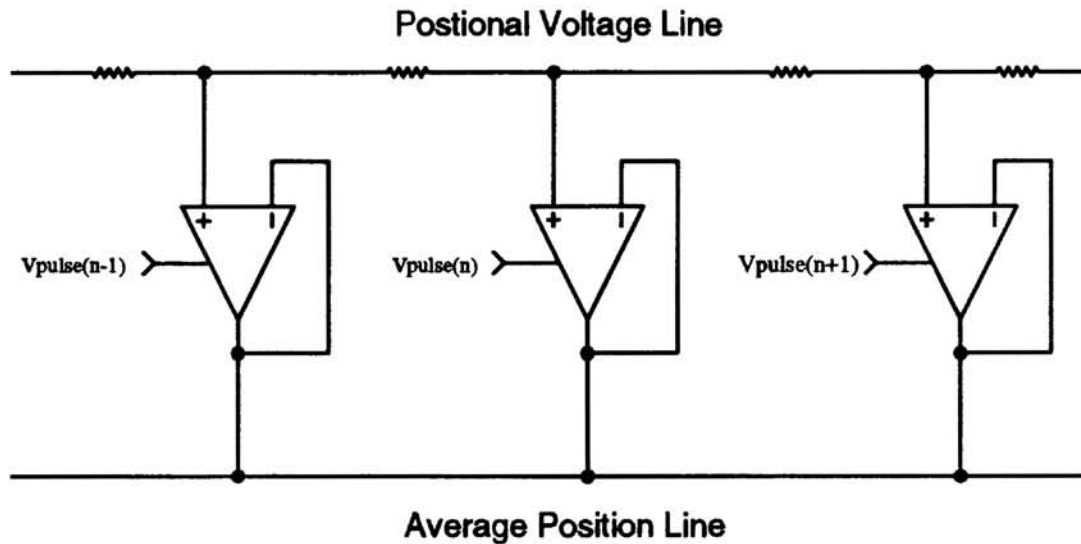

Figure 5: Circuit to determine object location

connected oscillators. This phase-locking could occur with oscillator frequencies set from 10 Hz to 4 KHz. A phase-locked group of oscillators would almost instantly split into two separate phase-locked groups with little temporal correlation between them when a connected chain of on oscillators was severed by turning off an oscillator in the middle of the original group. Mismatch in the transconductances of the oscillator transistors provided easy desynchronization.

Position tracking was measured by examining the resistive-line aggregator output during the time a certain phase-locked group of oscillators was pulsing. When multiple phase-locked groups of oscillators were active, it was still quite easy to make out the positional PAM voltage associated with each group by triggering an oscilloscope off of the pulsing output of an oscillator in that group. While there are occasional instances of two or more groups pulsing at the same time, if the duty cycle of the spiking oscillator is kept relatively small, there is little interference on average.

## 6    Discussion

It is becoming obvious that oscillation and synchronization phenomena in cortex may play an important role in neural information processing. In addition to striate cortex, the olfactory bulb also has oscillatory neural circuits which may be important in neural information processing (Freeman, 1988). It has been suggested that temporal correlation may be used for pattern segmentation in associative memories (Wang, 1990), and correlations between multiple oscillators may be used for storing time intervals (Miall, 1989).

We have described a circuit which performs Comparator Model phase-locking. The distributed and partitionable qualities of this circuit make it attractive as a possible physiological model. The PAM representation of object position shows one way that connectivity requirements can be minimized for communication in a neuromorphic

system. The chip has been fabricated using subthreshold CMOS technology, and thus uses little power.

## Acknowledgements

The authors are pleased to acknowledge helpful discussion with C. Koch and J. Lazzaro. Chip fabrication was provided by the MOSIS service.

## References

P. Baldi & R. Meir. (1990) Computing with arrays of coupled oscillators: an application to preattentive texture discrimination. *Neural Computation* **2**, 458–471.

F. Crick & C. Koch. (1990) Towards a neurobiological theory of consciousness. *Seminars in the Neurosciences* **2**, 263–275.

R. Eckhorn, H. J. Reitboek, M. Arndt & P. Dicke. (1990) Feature linking via synchronization among distributed assemblies: simulations of results from cat visual cortex. *Neural Computation* **2**, 293–307.

W. J. Freeman, Y. Yao, & B. Burke. (1988) Central pattern generating and recognizing in olfactory bulb: a correlation learning rule. *Neural Networks* **1**, 277–288.

C. M. Gray & W. Singer. (1987) Stimulus-specific neuronal oscillations in the cat visual cortex: A cortical functional unit. *Soc. Neurosci. Abstr.* **13**(404.3)

C. M. Gray, P. König, A. K. Engel & W. Singer. (1989) Oscillatory responses in cat visual cortex exhibit inter-columnar synchronization which reflects global stimulus properties. *Nature (London)* **338**, 334–337.

D. H. Hubel. (1988) *Eye, Brain, and Vision.* New York, NY: Scientific American Library.

D. M. Kammen, C. Koch & P. J. Holmes. (1990) Collective oscillations in the visual cortex. In D. S. Touretzky (ed.) *Advances in Neural Information Processing Systems 2.* San Mateo, CA: Morgan Kaufman Publishers.

C. A. Mead. (1989)*Analog VLSI and Neural Systems.* Reading, MA: Addison-Wesley.

M. A. Maher, S. P. Deweerth, M. A. Mahowald & C. A. Mead. (1989) Implementing neural architectures using analog VLSI circuits. *IEEE Trans. Circ. Sys.* **36**, 643–652.

C. Miall. (1989) The storage of time intervals using oscillating neurons. *Neural Computation* **1**, 359–371.

C. von der Malsburg. (1986) A neural cocktail-party processor. *Biological Cybernetics.* **54**,29-40.

D. Wang, J. Buhmann & C. von der Malsburg. (1990) Pattern segmentation in associative memory. *Neural Computation* **2**, 95–106.
